# Using Random Forests in the Structured Language Model

**Peng Xu  and  Frederick Jelinek**
Center for Language and Speech Processing
Department of Electrical and Computer Engineering
The Johns Hopkins University
{xp,jelinek}@jhu.edu

## Abstract

In this paper, we explore the use of Random Forests (RFs) in the structured language model (SLM), which uses rich syntactic information in predicting the next word based on words already seen. The goal in this work is to construct RFs by randomly growing Decision Trees (DTs) using syntactic information and investigate the performance of the SLM modeled by the RFs in automatic speech recognition.

RFs, which were originally developed as classifiers, are a combination of decision tree classifiers. Each tree is grown based on random training data sampled independently and with the same distribution for all trees in the forest, and a random selection of possible questions at each node of the decision tree. Our approach extends the original idea of RFs to deal with the data sparseness problem encountered in language modeling.

RFs have been studied in the context of $n$-gram language modeling and have been shown to generalize well to unseen data. We show in this paper that RFs using syntactic information can also achieve better performance in both perplexity (PPL) and word error rate (WER) in a large vocabulary speech recognition system, compared to a baseline that uses Kneser-Ney smoothing.

## 1   Introduction

In many systems dealing with speech or natural language, such as Automatic Speech Recognition and Statistical Machine Translation, a language model is a crucial component for searching in the often prohibitively large hypothesis space. Most state-of-the-art systems use $n$-gram language models, which are simple and effective most of the time. Many smoothing techniques that improve language model probability estimation have been proposed and studied in the $n$-gram literature  [1]. There has so far been work in exploring Decision Tree (DT) language models [2, 3], which attempt to cluster similar histories together to achieve better probability estimation. However, the results were negative [3]: decision tree language models failed to improve upon the baseline $n$-gram models with the same order $n$.

Random Forest (RF) language models, which are generalizations of DT language models, have been recently applied to word $n$-grams [4]. DT growing is randomized to construct

RFs efficiently. Once constructed, the RFs function as a randomized history clustering, which helps in dealing with the data sparseness problem. In general, the weakness of some trees can be compensated for by other trees. The collective contribution of all DTs in an RF $n$-gram model results in significant improvements in both perplexity (PPL) and word error rate (WER) in a large vocabulary speech recognition system.

Language models can also be improved with better representations of the history. Recent efforts have studied various ways of using information from a longer history span than that usually captured by normal $n$-gram language models, as well as ways of using syntactic information that is not available to the word-based $n$-gram models [5, 6, 7]. All these language models are based on stochastic parsing techniques that build up parse trees for the input word sequence and condition the generation of words on syntactic and lexical information available in the parse trees. Since these language models capture useful hierarchical characteristics of language, they can improve PPL and WER significantly for various tasks. However, due to the $n$-gram nature of the components of the syntactic language models, the data sparseness problem can be severe.

In order to reduce the data sparseness problem for using rich syntactic information in the context, we study the use of RFs in the structured language model (SLM) [5]. Our results show that although the components of the SLM have high order $n$-grams, our RF approach can still achieve better performance, reducing both the perplexity (PPL) and word error rate (WER) in a large vocabulary speech recognition system compared to a Kneser-Ney smoothing baseline.

## 2 Basic Language Modeling

The purpose of a language model is to estimate the probability of a word string. Let $W$ denote a string of $N$ words, that is, $W = w_1, w_2, \ldots, w_N$. Then, by the chain rule of probability, we have

$$P(W) = P(w_1) \times \prod_{i=2}^{N} P(w_i | w_1, \ldots, w_{i-1}). \tag{1}$$

In order to estimate the probabilities $P(w_i | w_1, \ldots, w_{i-1})$, we need a training corpus consisting of a large number of words. However, in any practical natural language system of even moderate vocabulary size, it is clear that the number of probabilities to be estimated and stored is prohibitively large. Therefore, histories $w_1, \ldots, w_{i-1}$ for a word $w_i$ are usually grouped into equivalence classes. The most widely used language models, $n$-gram language models, use the identities of the last $n - 1$ words as equivalence classes. In an $n$-gram model, we then have

$$P(W) = P(w_1) \times \prod_{i=2}^{N} P(w_i | w_{i-n+1}^{i-1}), \tag{2}$$

where we have used $w_{i-n+1}^{i-1}$ to denote the word sequence $w_{i-n+1}, \ldots, w_{i-1}$.

If we could handle unlimited amounts of training data, the maximum likelihood (ML) estimate of $P(w_i | w_{i-n+1}^{i-1})$ would be the best:

$$P(w_i | w_{i-n+1}^{i-1}) = \frac{C(w_{i-n+1}^{i})}{C(w_{i-n+1}^{i-1})}, \tag{3}$$

where $C(w_{i-n+1}^{i})$ is the number of times the string $w_{i-n+1}, \ldots, w_i$ is seen in the training data.

### 2.1 Language Model Smoothing

An $n$-gram model when $n = 3$ is called a trigram model. For a vocabulary of size $|V| = 10^4$, there are $|V|^3 = 10^{12}$ trigram probabilities to be estimated. For any training data of a manageable size, many of the probabilities will be zero if the ML estimate is used.

In order to solve this problem, many smoothing techniques have been studied (see [1] and the references therein). Smoothing adjusts the ML estimates to produce more accurate probabilities and to assign nonzero probabilities to any word string. Details about various smoothing techniques will not be presented in this paper, but we will outline a particular way of smoothing, namely interpolated Kneser-Ney smoothing [8], for later reference.

Interpolated Kneser-Ney smoothing assumes the following form:

$$P_{KN}(w_i|w_{i-n+1}^{i-1}) = \frac{max(C(w_{i-n+1}^i) - D, 0)}{C(w_{i-n+1}^{i-1})}$$
$$+ \lambda(w_{i-n+1}^{i-1}) P_{KN}(w_i|w_{i-n+2}^{i-1}), \tag{4}$$

where $D$ is a discounting constant and $\lambda(w_{i-n+1}^{i-1})$ is the interpolation weight for the lower order probabilities ($(n-1)$-gram). The discount constant is often estimated using the leave-one-out method, leading to the approximation $D = \frac{n_1}{n_1 + 2n_2}$, where $n_1$ is the number of $n$-grams with count one and $n_2$ is the number of $n$-grams with count two. To ensure that the probabilities sum to one, we have

$$\lambda(w_{i-n+1}^{i-1}) = \frac{D \sum_{w_i:C(w_{i-n+1}^i)>0} 1}{C(w_{i-n+1}^{i-1})}.$$

The lower order probabilities in interpolated Kneser-Ney smoothing can be estimated as (assuming ML estimation):

$$P_{KN}(w_i|w_{i-n+2}^{i-1}) = \frac{\sum_{w_{i-n+1}:C(w_{i-n+1}^i)>0} 1}{\sum_{w_{i-n+1},w_i:C(w_{i-n+1}^i)>0} 1}. \tag{5}$$

Note that the lower order probabilities are usually recursively smoothed using Equation 4.

## 2.2 Language Model Evalution

A commonly used task-independent quality measure for a given language model is related to the cross-entropy of the underlying model and is referred to as *perplexity* (PPL):

$$PPL = exp(-1/N \sum_{i=1}^N \log[P(w_i|w_1^{i-1})]), \tag{6}$$

where $w_1, \ldots, w_N$ is the test text that consists of $N$ words.

For different tasks, there are different task-dependent quality measures of language models. For example, in an automatic speech recognition system, the performance is usually measured by word error rate (WER).

## 3 The Structured Language Model (SLM)

The SLM uses rich syntactic information beyond regular word $n$-grams to improve language model quality. An extensive presentation of the SLM can be found in Chelba and Jelinek, 2000 [5]. The model assigns a probability $P(W, T)$ to every sentence $W$ and every possible binary parse $T$. The terminals of $T$ are the words of $W$ with POS tags, and the nodes of $T$ are annotated with phrase headwords and non-terminal labels. Let $W$ be

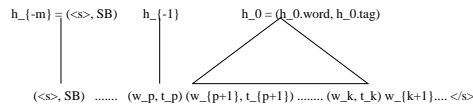

Figure 1: A word-parse $k$-prefix

a sentence of length $n$ words to which we have prepended the sentence beginning marker `<s>` and appended the sentence end marker `</s>` so that $w_0 =$`<s>` and $w_{n+1} =$`</s>`. Let $W_k = w_0 \ldots w_k$ be the *word k-prefix* of the sentence — the words from the beginning of the sentence up to the current position $k$ — and $W_k T_k$ the *word-parse k-prefix*. Figure 1 shows a word-parse $k$-prefix; `h_0, .., h_{-m}` are the *exposed heads*, each head being a pair (headword, non-terminal label), or (word, POS tag) in the case of a root-only tree. The exposed heads at a given position $k$ in the input sentence are a function of the word-parse $k$-prefix [5].

The joint probability $P(W, T)$ of a word sequence $W$ and a complete parse $T$ comes from contributions of three components: WORD-PREDICTOR, TAGGER and CONSTRUCTOR. The SLM works in the following way: first, the WORD-PREDICTOR predicts a word based on the word-parse prefix; the TAGGER then assigns a POS tag to the predicted word based on the word itself and the word-parse prefix; the CONSTRUCTOR takes a series of actions each of which turns a parse prefix into a new parse prefix (the series of actions ends with a *NULL* action which tells the WORD-PREDICTOR to predict the next word). Details about the three components can be found in [5]. Each of the three components can be seen as an $n$-gram model and can be estimated independently because of the product form of the joint probability. They are parameterized (approximated) as follows:

$$P(w_k|W_{k-1}T_{k-1}) = P(w_k|h_0.tag,h_0.word,h_{-1}.tag,h_{-1}.word), \qquad (7)$$

$$P(t_k|w_k,W_{k-1}T_{k-1}) = P(t_k|w_k,h_0.tag,h_{-1}.tag), \qquad (8)$$

$$P(p_i^k|W_{k-1}T_{k-1},w_k,t_k,p_1^k\ldots p_{i-1}^k) = P(p_i^k|h_0.tag,h_{-1}.tag,h_{-2}.tag,h_0.word,h_{-1}.word), \quad (9)$$

where $p_i^k$ is the $i^{th}$ CONSTRUCTOR action after the $k^{th}$ word and POS tag have been predicted. Since the number of parses for a given word prefix $W_k$ grows exponentially with $k$, $|\{T_k\}| \sim O(2^k)$, the state space of our model is huge even for relatively short sentences. Thus we must use a search strategy that prunes the allowable parse set. One choice is a synchronous multi-stack search algorithm [5] which is very similar to a beam search.

The language model probability assignment for the word at position $k + 1$ in the input sentence is made using:

$$P_{SLM}(w_{k+1}|W_k) = \sum_{T_k \in S_k} P(w_{k+1}|W_k T_k) \cdot \rho(W_k,T_k),$$

$$\rho(W_k,T_k) = P(W_k T_k)/\sum_{T_k \in S_k} P(W_k T_k), \qquad (10)$$

which ensures a proper probability normalization over strings of words, where $S_k$ is the set of all parses present in the stacks at the current stage $k$ and $P(W_k T_k)$ is the joint probability of word-parse prefix $W_k T_k$.

Each model component —WORD-PREDICTOR, TAGGER, CONSTRUCTOR— is estimated independently from a set of parsed sentences after undergoing headword percolation and binarization (see details in [5]).

## 4 Using Random Forests in the Structured Language Model

### 4.1 Random Forest $n$-gram Modeling

A Random Forest (RF) $n$-gram model is a collection of randomly constructed decision tree (DT) $n$-gram models. Unlike RFs in classification and regression tasks [9, 10, 11], RFs are used in language modeling to deal with the data sparseness problem [4]. Therefore, the training data is not randomly sampled for each DT. Figure 2 shows the algorithm **DT-Grow** and **Node-Split** used for generating random DT language models.

We define a *position* in the history as the distance between a word in the history and the predicted word. The randomization is carried out in two places: a random selection of

**Algorithm DT-Grow**
**Input**: counts for training and heldout data

**Initialize**: Create a root node containing all histories in the training data and put it in set $\Phi$

**While** $\Phi$ is not empty
1. Get a node $p$ from $\Phi$
2. **If Node-Split($p$) is** successful, eliminate $p$ from $\Phi$ and put the two children of $p$ in $\Phi$

**Foreach** internal node $p$ in the tree
1. $L_p^H \leftarrow$ normalized likelihood of heldout data associated with $p$, using training data statistics in $p$
2. Get the set of leaves $\mathcal{P}$ rooted in $p$
3. $L_\mathcal{P}^H \leftarrow$ normalized likelihood of heldout data associated with all leaves in $\mathcal{P}$, using training data statistics in the corresponding leaves
4. if $L_\mathcal{P}^H - L_p^H < 0$, prune the subtree rooted in $p$

**Output**: a Decistion Tree language model

**Algorithm Node-Split($p$)**
**Input**: node $p$ and training data associated

**Initialize**: Randomly select a subset of positions $I$ in the history

**Foreach** position $i$ in $I$
1. Group all histories into basic elements $\beta(v)$
2. Randomly split the elements $\beta(v)$ into sets $\mathcal{L}$ and $\mathcal{R}$
3. **While** there are elements moved, **Do**

   (a) Move each element from $\mathcal{L}$ to $\mathcal{R}$ if the move results in positive gain in training data likelihood

   (b) Move each element from $\mathcal{R}$ to $\mathcal{L}$ if the move results in positive gain in training data likelihood

Select the position from $I$ that results in the largest gain

**Output**: a split $\mathcal{L}$ and $\mathcal{R}$, or failure if the largest gain is not positive

Figure 2: The algorithm **DT-Grow** and **Node-Split**

positions in the history and an initial random split of basic elements. Since our splitting criterion is to maximize the log-likelihood of the training data, each split uses only statistics (from training data) associated with the node under consideration. Smoothing is not needed in the splitting and we can use a fast exchange algorithm [12] in **Node-Split**. Given a position $i$ in the history, $\beta(v)$ is defined to be the set of histories belonging to the node $p$, such that they all have word $v$ at position $i$. It is clear that for every position $i$ in the history, the union $\cup_v \beta(v)$ is all histories in the node $p$.

In **DT-Grow**, after a DT is fully grown, we use some heldout data to prune it. Pruning is done in such a way that we maximize the likelihood of the heldout data, where smoothing is applied according to Equation 4:

$$P_{DT}(w_i|\Phi_{DT}(w_{i-n+1}^{i-1})) = \frac{max(C(w_i,\Phi_{DT}(w_{i-n+1}^{i-1}))-D,0)}{C(\Phi_{DT}(w_{i-n+1}^{i-1}))} \\ + \lambda(\Phi_{DT}(w_{i-n+1}^{i-1}))P_{KN}(w_i|w_{i-n+2}^{i-1}) \quad (11)$$

where $\Phi_{DT}(\cdot)$ is one of the DT nodes the history can be mapped to and $P_{KN}(w_i|w_{i-n+2}^{i-1})$ is as defined in Equation 5. This pruning is similar to the pruning strategy used in CART [13].

Once we get the DTs, we only use the leaf nodes as equivalence classes of histories. If a new history is encountered, it is very likely that we will not be able to place it at a leaf node in the DT. In this case, $\lambda(\Phi_{DT}(w_{i-n+1}^{i-1})) = 1$ in Equation 11 and we simply use $P_{KN}(w_i|w_{i-n+2}^{i-1})$ to get the probabilities.

The randomized version of the DT growing algorithm can be run many times and finally we will get a collection of randomly grown DTs: a Random Forest (RF). Since each DT is a smoothed language model, we simply aggregate all DTs in our RF to get the RF language

model. Suppose we have $M$ randomly grown DTs, $DT_1, \ldots, DT_M$. In the $n$-gram case, the RF language model probabilities can be computed as:

$$P_{RF}(w_i|w_{i-n+1}^{i-1}) = \frac{1}{M} \sum_{j=1}^{M} P_{DT_j}(w_i|\Phi_{DT_j}(w_{i-n+1}^{i-1})) \tag{12}$$

where $\Phi_{DT_j}(w_{i-n+1}^{i-1})$ maps the history $w_{i-n+1}^{i-1}$ to a leaf node in $DT_j$. If $w_{i-n+1}^{i-1}$ can not be mapped to a leaf node in some DT, we back-off to the lower order KN probability as mentioned earlier.

It can be shown by the Law of Large Numbers that the probability in Equation 12 converges as the number of DTs grows. It converges to $E_{\mathcal{T}}\left[P_{\mathcal{T}}(w_i|\Phi_{\mathcal{T}}(w_{i-n+1}^{i-1}))\right]$ where $\mathcal{T}$ is a random variable representing the random DTs. The advantage of the RF approach over the KN smoothing lies in the fact that different DTs have different weaknesses and strengths for word prediction. As the number of trees grows, the weakness of some trees can be compensated for by some other trees. This advantage and the convergence have been shown experimentally in [4].

## 4.2  Using RFs in the SLM

Since the three model components in the SLM as in Equation 7-9 can be estimated independently, we can construct an RF for each component using the algorithm **DT-Grow** in the previous section. The only difference is that we will have different $n$-gram orders and different items in the history for each model.

Ideally, we would like to use RFs for each component in the SLM. However, due to the nature of the SLM, there are difficulties. The SLM uses a synchronous multi-stack search algorithm to dynamically construct stacks and compute the language model probabilities as in Equation 10. If we use RFs for all components, we need to load all DTs in the RFs into memory at runtime. This is impractical for RFs of any reasonable size.

There is a different approach that can take advantage of the randomness in the RFs. Suppose we have $M$ randomly grown DTs, $DT_1^a, \ldots, DT_M^a$ for each component $a$ of the SLM, where $a \in \{P, T, C\}$ for WORD-PREDICTOR, TAGGER and CONSTRUCTOR, respectively. The DTs are grouped into $M$ triples $\{DT_j^P, DT_j^T, DT_j^C\}$ $j = 1, \ldots, M$. We calculate the joint probability $P(W, T)$ for the $j^{th}$ DT triple according to:

$$
\begin{aligned}
P_j(W,T) = \quad & \prod_{k=1}^{n+1} [P_{DT_j^P}(w_k|W_{k-1}T_{k-1}) \cdot P_{DT_j^T}(t_k|W_{k-1}T_{k-1},w_k) \cdot \\
& \prod_{i=1}^{N_k} P_{DT_j^C}(p_i^k|W_{k-1}T_{k-1},w_k,t_k,p_1^k...p_{i-1}^k)].
\end{aligned} \tag{13}
$$

Then, the language model probability assignment for the $j^{th}$ DT triple is made using:

$$
\begin{aligned}
P_j(w_{k+1}|W_k) &= \sum_{T_k^j \in S_k^j} P_{DT_j^P}(w_{k+1}|W_k T_k^j) \cdot \rho_j(W_k, T_k^j), \\
\rho_j(W_k, T_k^j) &= P_j(W_k T_k^j) / \sum_{T_k^j \in S_k^j} P_j(W_k T_k^j),
\end{aligned} \tag{14}
$$

which is achieved by running the synchronous multi-stack algorithm using the $j^{th}$ DT triple as a model. Finally, after the SLM is run $M$ times, the RF language model probability is an average of the probabilities above:

$$P_{RF}(w_{k+1}|W_k) = \frac{1}{M} \sum_{j=1}^{M} P_j(w_{k+1}|W_k). \tag{15}$$

The triple $\{DT_j^P, DT_j^T, DT_j^C\}$ can be considered as a single DT in which the root node has three children corresponding to the three root nodes of $DT_j^P, DT_j^T$ and $DT_j^C$. The root node of this DT asks the question: Which model component does the history belong to? According to the answer, we can proceed to one of the three children nodes (one of the three components, in fact). Since the multi-stack search algorithm is deterministic given the DT, the probability in Equation 15 can be shown to converge.

# 5 Experiments

## 5.1 Perplexity (PPL)

We have used the UPenn Treebank portion of the WSJ corpus to carry out our experiments. The UPenn Treebank contains 24 sections of hand-parsed sentences, for a total of about one million words. We used section 00-20 for training our models, section 21-22 as heldout data for pruning the DTs, and section 23-24 to test our models. Before carrying out our experiments, we normalized the text in the following ways: numbers in arabic form were replaced by a single token "N", punctuation was removed, all words were mapped to lower case, extra information in the parse trees was ignored, and, finally, traces were ignored. The word vocabulary contains 10k words including a special token for unknown words. There are 40 items in the part-of-speech set and 54 items in the non-terminal set, respectively.

The three components in the SLM were treated independently during training. We trained an RF for each component and each RF contained 100 randomly grown DTs. The baseline SLM used KN smoothing (KN-SLM). The 100 probability sequences from the 100 triples were aggregated to get the final PPL. The results are shown in Table 1. We also interpolated the SLM with the KN-trigram to get further improvements. The interpolation weight $\alpha$ in Table 1 is on KN-trigram. The RF-SLM achieved a 10.9% and a 7.5% improvement over the KN-SLM, before and after interpolation with KN-trigram, respectively. Compared to the improvements reported in [4] (10.5% from RF-trigram to KN-trigram), the RF-SLM achieved greater improvement by using syntactic information. Figure 3 shows the convergence of the PPL as the number of DTs grows from 1 to 100.

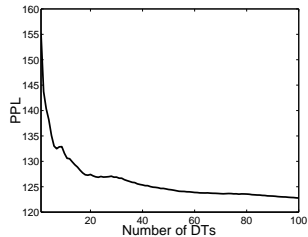

Figure 3: PPL convergence

| Model | $\alpha$=0.0 | $\alpha$=0.4 | $\alpha$=1.0 |
|---|---|---|---|
| KN-SLM | 137.9 | 127.2 | 145.0 |
| RF-SLM | 122.8 | 117.6 | 145.0 |
| Gain | 10.9% | 7.5% | - |

Table 1: PPL comparison between KN-SLM and RF-SLM, interpolated with KN-trigram

## 5.2 Word Error Rate by $N$-best Re-scoring

To test our RF modeling approach in the context of speech recognition, we evaluated the models in the WSJ DARPA'93 HUB1 test setup. The size of the test set is 213 utterances, 3446 words. The 20k word open vocabulary and baseline 3-gram model are the standard ones provided by NIST and LDC — see [5] for details. The $N$-best lists were generated using the standard 3-gram model trained on 40M words of WSJ. The $N$-best size was at most 50 for each utterance, and the average size was about 23. For the KN-SLM and RF-SLM, we used 20M words automatically parsed, binarized and enriched with headwords and NT/POS tag information. As the size of RF-SLM becomes very large, we only used RF for the WORD-PREDICTOR component (RF-SLM-P). The other two components used KN smoothing. The results are reported in Table 2.

| Model | $\alpha$=0.0 | $\alpha$=0.2 | $\alpha$=0.4 | $\alpha$=0.6 | $\alpha$=0.8 |
|---|---|---|---|---|---|
| KN-SLM | 12.8 | 12.5 | 12.6 | 12.7 | 12.7 |
| RF-SLM-P | 11.9 | 12.2 | 12.3 | 12.3 | 12.6 |

Table 2: N-best rescoring WER results

For purpose of comparison, we interpolated all models with the KN-trigram built from

40M words at different level of interpolation weights $\alpha$ (on KN-trigram). However, it is the $\alpha = 0.0$ column that is the most interesting. We can see that the RF approach improved over the regular KN approach with an absolute WER reduction of 0.9%.

## 6 Conclusions

Based on the idea of Random Forests in classification and regression, we developed algorithms for constructing and using Random Forests in language modeling. In particular, we applied this new probability estimation technique to the Structured Language Model, in which there are three model components that can be estimated independently. The independently constructed Random Forests can be considered as a more general single Random Forest, which ensures the convergence of the probabilities as the number of Decision Trees grows. The results on a large vocabulary speech recognition system show that we can achieve significant reduction in both perplexity and word error rate, compared to a baseline using Kneser-Ney smoothing.

## References

[1] Stanley F. Chen and Joshua Goodman, "An empirical study of smoothing techniques for language modeling," Tech. Rep. TR-10-98, Computer Science Group, Harvard University, Cambridge, Massachusetts, 1998.

[2] L. Bahl, P. Brown, P. de Souza, and R. Mercer, "A tree-based statistical language model for natural language speech recognition," in *IEEE Transactions on Acoustics, Speech and Signal Processing*, July 1989, vol. 37, pp. 1001–1008.

[3] Gerasimos Potamianos and Frederick Jelinek, "A study of n-gram and decision tree letter language modeling methods," *Speech Communication*, vol. 24(3), pp. 171–192, 1998.

[4] Peng Xu and Frederick Jelinek, "Random forests in language modeling," in *Proceedings of the 2004 Conference on Empirical Methods in Natural Language Processing*, Barcelona, Spain, July 2004.

[5] Ciprian Chelba and Frederick Jelinek, "Structured language modeling," *Computer Speech and Language*, vol. 14, no. 4, pp. 283–332, October 2000.

[6] Eugene Charniak, "Immediate-head parsing for language models," in *Proceedings of the 39th Annual Meeting and 10th Conference of the European Chapter of ACL*, Toulouse, France, July 2001, pp. 116–123.

[7] Brian Roark, *Robust Probabilistic Predictive Syntactic Processing: Motivations, Models and Applications*, Ph.D. thesis, Brown University, Providence, RI, 2001.

[8] Reinhard Kneser and Hermann Ney, "Improved backing-off for m-gram language modeling," in *Proceedings of the IEEE International Conference on Acoustics, Speech, and Signal Processing*, 1995, vol. 1, pp. 181–184.

[9] Y. Amit and D. Geman, "Shape quantization and recognition with randomized trees," *Neural Computation*, , no. 9, pp. 1545–1588, 1997.

[10] Leo Breiman, "Random forests," Tech. Rep., Statistics Department, University of California, Berkeley, Berkeley, CA, 2001.

[11] T.K. Ho, "The random subspace method for constructing decision forests," *IEEE Trans. on Pattern Analysis and Machine Intelligence*, vol. 20, no. 8, pp. 832–844, 1998.

[12] S. Martin, J. Liermann, and H. Ney, "Algorithms for bigram and trigram word clustering," *Speech Communication*, vol. 24(3), pp. 171–192, 1998.

[13] L. Breiman, J.H. Friedman, R.A. Olshen, and C.J. Stone, *Classification and Regression Trees*, Chapman and Hall, New York, 1984.
